# Boosting Classifier Cascades

**Mohammad J. Saberian**
Statistical Visual Computing Laboratory,
University of California, San Diego
La Jolla, CA 92039
saberian@ucsd.edu

**Nuno Vasconcelos**
Statistical Visual Computing Laboratory,
University of California, San Diego
La Jolla, CA 92039
nuno@ucsd.edu

## Abstract

The problem of optimal and automatic design of a detector cascade is considered. A novel mathematical model is introduced for a cascaded detector. This model is analytically tractable, leads to recursive computation, and accounts for both classification and complexity. A boosting algorithm, FCBoost, is proposed for fully automated cascade design. It exploits the new cascade model, minimizes a Lagrangian cost that accounts for both classification risk and complexity. It searches the space of cascade configurations to automatically determine the optimal number of stages and their predictors, and is compatible with bootstrapping of negative examples and cost sensitive learning. Experiments show that the resulting cascades have state-of-the-art performance in various computer vision problems.

## 1 Introduction

There are many applications where a classifier must be designed under computational constraints. One problem where such constraints are extreme is that of object detection in computer vision. To accomplish tasks such as face detection, the classifier must process thousands of examples per image, extracted from all possible image locations and scales, at a rate of several images per second. This problem has been the focus of substantial attention since the introduction of the detector cascade architecture by Viola and Jones (VJ) in [13]. This architecture was used to design the first real time face detector with state-of-the-art classification accuracy. The detector has, since, been deployed in many practical applications of broad interest, e.g. face detection on low-complexity platforms such as cameras or cell phones. The outstanding performance of the VJ detector is the result of 1) a cascade of simple to complex classifiers that reject most non-faces with a few machine operations, 2) learning with a combination of boosting and Haar features of extremely low complexity, and 3) use of bootstrapping to efficiently deal with the extremely large class of non-face examples.

While the resulting detector is fast and accurate, the process of designing a cascade is not. In particular, VJ did not address the problem of how to automatically determine the optimal cascade configuration, e.g. the numbers of cascade stages and weak learners per stage, or even how to design individual stages so as to guarantee optimality of the cascade as a whole. In result, extensive manual supervision is required to design cascades with good speed/accuracy trade off. This includes trial-and-error tuning of the false positive/detection rate of each stage, and of the cascade configuration. In practice, the design of a good cascade can take up several weeks. This has motivated a number of enhancements to the VJ training procedure, which can be organized into three main areas: 1) enhancement of the boosting algorithms used in cascade design, e.g. cost-sensitive variations of boosting [12, 4, 8], float Boost [5] or KLBoost [6], 2) post processing of a learned cascade, by adjusting stage thresholds, to improve performance [7], and 3) specialized cascade architectures which simplify the learning process, e.g. the embedded cascade (ChainBoost) of [15], where each stage contains all weak learners of previous stages. These enhancements do not address the fundamental limitations of the VJ design, namely how to guarantee overall cascade optimality.

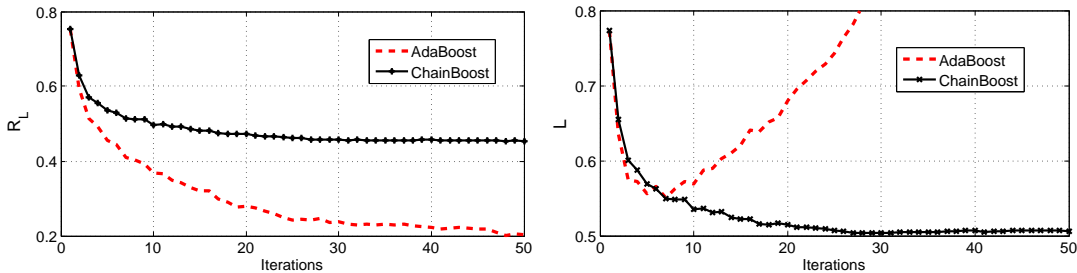

Figure 1: Plots of $R_L$ (left) and $\mathcal{L}$ (right) for detectors designed with AdaBoost and ChainBoost.

More recently, various works have attempted to address this problem [9, 8, 1, 14, 10]. However, the proposed algorithms still rely on sequential learning of cascade stages, which is suboptimal, sometimes require manual supervision, do not search over cascade configurations, and frequently lack a precise mathematical model for the cascade. In this work, we address these problems, through two main contributions. The first is a mathematical model for a detector cascade, which is analytically tractable, accounts for both classification and complexity, and is amenable to recursive computation. The second is a boosting algorithm, FCBoost, that exploits this model to solve the cascade learning problem. FCBoost solves a Lagrangian optimization problem, where the classification risk is minimized under complexity constraints. The risk is that of the entire cascade, which is learned holistically, rather than through sequential stage design, and FCBoost determines the optimal cascade configuration automatically. It is also compatible with bootstrapping and cost sensitive boosting extensions, enabling efficient sampling of negative examples and explicit control of the false positive/detection rate trade off. An extensive experimental evaluation, covering the problems of face, car, and pedestrian detection demonstrates its superiority over previous approaches.

## 2 Problem Definition

A binary classifier $h(x)$ maps an example $x$ into a class label $y \in \{-1, 1\}$ according to $h(x) = sign[f(x)]$, where $f(x)$ is a continuous-valued predictor. Optimal classifiers minimize a risk

$$R_L(f) = E_{X,Y}\{L[y, f(x)]\} \simeq \frac{1}{|S_t|} \sum_i L[y_i, f(x_i)] \tag{1}$$

where $S_t = \{(x_1, y_1), \ldots, (x_n, y_n)\}$ is a set of training examples, $y_i \in \{1, -1\}$ the class label of example $x_i$, and $L[y, f(x)]$ a loss function. Commonly used losses are upper bounds on the zero-one loss, whose risk is the probability of classification error. Hence, $R_L$ is a measure of classification accuracy. For applications with computational constraints, optimal classifier design must also take into consideration the classification complexity. This is achieved by defining a computational risk

$$R_C(f) = E_{X,Y}\{L_C[y, \mathcal{C}(f(x))]\} \simeq \frac{1}{|S_t|} \sum_i L_C[y_i, \mathcal{C}(f(x_i))] \tag{2}$$

where $\mathcal{C}(f(x))$ is the complexity of evaluating $f(x)$, and $L_C[y, \mathcal{C}(f(x))]$ a loss function that encodes the cost of this operation. In most detection problems, targets are rare events and contribute little to the overall complexity. In this case, which we assume throughout this work, $L_C[1, \mathcal{C}(f(x))] = 0$ and $L_C[-1, \mathcal{C}(f(x))]$ is denoted $L_C[\mathcal{C}(f(x))]$. The computational risk is thus

$$R_C(f) \approx \frac{1}{|S_t^-|} \sum_{x_i \in S_t^-} L_C[\mathcal{C}(f(x_i))]. \tag{3}$$

where $S_t^-$ contains the negative examples of $S_t$. Usually, more accurate classifiers are more complex. For example in boosting, where the decision rule is a combination of weak rules, a finer approximation of the classification boundary (smaller error) requires more weak learners and computation.

Optimal classifier design under complexity constraints is a problem of constrained optimization, which can be solved with Lagrangian methods. These minimize a Lagrangian

$$\mathcal{L}(f; S_t) = \frac{1}{|S_t|} \sum_{x_i \in S_t} L[y_i, f(x_i)] + \frac{\eta}{|S_t^-|} \sum_{x_i \in S_t^-} L_C[\mathcal{C}(f(x_i))], \tag{4}$$

where $\eta$ is a Lagrange multiplier, which controls the trade-off between error rate and complexity. Figure 1 illustrates this trade-off, by plotting the evolution of $R_L$ and $\mathcal{L}$ as a function of the boosting iteration, for the AdaBoost algorithm [2]. While the risk always decreases with the addition of weak learners, this is not true for the Lagrangian. After a small number of iterations, the gain in accuracy does not justify the increase in classifier complexity. The design of classifiers under complexity constraints has been addressed through the introduction of detector cascades. A detector cascade $\mathcal{H}(x)$ implements a sequence of binary decisions $h_i(x), i = 1 \ldots m$. An example $x$ is declared a target ($y = 1$) if and only if it is declared a target by all stages of $\mathcal{H}$, i.e. $h_i(x) = 1, \forall i$. Otherwise, the example is rejected. For applications where the majority of examples can be rejected after a small number of cascade stages, the average classification time is very small. However, the problem of designing an optimal detector cascade is still poorly understood. A popular approach, known as *ChainBoost* or *embedded cascade* [15], is to 1) use standard boosting algorithms to design a detector, and 2) insert a rejection point after each weak learner. This is simple to implement, and creates a cascade with as many stages as weak learners. However, the introduction of the intermediate rejection points, *a posteriori* of detector design, sacrifices the risk-optimality of the detector. This is illustrated in Figure 1, where the evolution of $R_L$ and $\mathcal{L}$ are also plotted for ChainBoost. In this example, $\mathcal{L}$ is monotonically decreasing, i.e. the addition of weak learners no longer carries a large complexity penalty. This is due to the fact that most negative examples are rejected in the earliest cascade stages. On the other hand, the classification risk is more than double that of the original boosted detector. It is not known how close ChainBoost is to optimal, in the sense of (4).

## 3 Classifier cascades

In this work, we seek the design of cascades that are provably optimal under (4). We start by introducing a mathematical model for a detector cascade.

### 3.1 Cascade predictor

Let $\mathcal{H}(x) = \{h_1(x), \ldots, h_m(x)\}$ be a cascade of $m$ detectors $h_i(x) = sgn[f_i(x)]$. To develop some intuition, we start with a two-stage cascade, $m = 2$. The cascade implements the decision rule

$$\mathcal{H}(\mathcal{F})(x) = sgn[\mathcal{F}(x)] \tag{5}$$

where

$$\mathcal{F}(x) = \mathcal{F}(f_1, f_2)(x) = \begin{cases} f_1(x) & \text{if } f_1(x) < 0 \\ f_2(x) & \text{if } f_1(x) \geq 0 \end{cases} \tag{6}$$

$$= f_1 u(-f_1) + u(f_1) f_2 \tag{7}$$

is denoted the *cascade predictor*, $u(.)$ is the step function and we omit the dependence on $x$ for notational simplicity. This equation can be extended to a cascade of $m$ stages, by replacing the predictor of the second stage, when $m = 2$, with the predictor of the remaining cascade, when $m$ is larger. Letting $\mathcal{F}_j = \mathcal{F}(f_j, \ldots, f_m)$ be the cascade predictor for the cascade composed of stages $j$ to $m$

$$\mathcal{F} = \mathcal{F}_1 = f_1 u(-f_1) + u(f_1) \mathcal{F}_2. \tag{8}$$

More generally, the following recursion holds

$$\mathcal{F}_k = f_k u(-f_k) + u(f_k) \mathcal{F}_{k+1} \tag{9}$$

with initial condition $\mathcal{F}_m = f_m$. In Appendix A, it is shown that combining (8) and (9) recursively leads to

$$\mathcal{F} = T_{1,m} + T_{2,m} f_m \tag{10}$$

$$= T_{1,k} + T_{2,k} f_k u(-f_k) + T_{2,k} \mathcal{F}_{k+1} u(f_k), \quad k < m. \tag{11}$$

with initial conditions $T_{1,0} = 0$, $T_{2,0} = 1$ and

$$T_{1,k+1} = T_{1,k} + f_k u(-f_k) T_{2,k}, \qquad T_{2,k+1} = T_{2,k} u(f_k). \tag{12}$$

Since $T_{1,k}, T_{2,k}$, and $\mathcal{F}_{k+1}$ do not depend on $f_k$, (10) and (11) make explicit the dependence of the cascade predictor, $\mathcal{F}$, on the predictor of the $k^{th}$ stage.

## 3.2 Differentiable approximation

Letting $\mathcal{F}(f_k + \epsilon g) = \mathcal{F}(f_1, .., f_k + \epsilon g, .. f_m)$, the design of boosting algorithms requires the evaluation of both $\mathcal{F}(f_k + \epsilon g)$, and the functional derivative of $\mathcal{F}$ with respect to each $f_k$, along any direction $g$

$$< \delta\mathcal{F}(f_k), g >= \frac{d}{d\epsilon}\mathcal{F}(f_k + \epsilon g)\bigg|_{\epsilon=0}.$$

These are straightforward for the last stage since, from (10),

$$\mathcal{F}(f_m + \epsilon g) = a^m + \epsilon b^m g, \quad < \delta\mathcal{F}(f_m), g >= b^m g, \tag{13}$$

where

$$a^m = T_{1,m} + T_{2,m}f_m = \mathcal{F}(f_m), \quad b^m = T_{2,m}. \tag{14}$$

In general, however, the right-hand side of (11) is non-differentiable, due to the $u(.)$ functions. A differentiable approximation is possible by adopting the classic sigmoidal approximation $u(x) \approx \frac{\tanh(\sigma x)+1}{2}$, where $\sigma$ is a relaxation parameter. Using this approximation in (11),

$$\mathcal{F} = \mathcal{F}(f_k) \quad = \quad T_{1,k} + T_{2,k}f_k(1 - u(f_k)) + T_{2,k}\mathcal{F}_{k+1}u(f_k) \tag{15}$$

$$\approx \quad T_{1,k} + T_{2,k}f_k + \frac{1}{2}T_{2,k}[\mathcal{F}_{k+1} - f_k][\tanh(\sigma f_k) + 1]. \tag{16}$$

It follows that

$$< \delta\mathcal{F}(f_k), g > \quad = \quad b^k g \tag{17}$$

$$b^k \quad = \quad \frac{1}{2}T_{2,k}\left\{[1 - \tanh(\sigma f_k)] + \sigma[\mathcal{F}_{k+1} - f_k][1 - \tanh^2(\sigma f_k)]\right\}. \tag{18}$$

$\mathcal{F}(f_k + \epsilon g)$ can also be simplified by resorting to a first order Taylor series expansion around $f_k$

$$\mathcal{F}(f_k + \epsilon g) \quad \approx \quad a^k + \epsilon b^k g \tag{19}$$

$$a^k \quad = \quad \mathcal{F}(f_k) = T_{1,k} + T_{2,k}\left\{f_k + \frac{1}{2}[\mathcal{F}_{k+1} - f_k][\tanh(\sigma f_k) + 1]\right\}. \tag{20}$$

## 3.3 Cascade complexity

In Appendix B, a similar analysis is performed for the computational complexity. Denoting by $\mathcal{C}(f_k)$ the complexity of evaluating $f_k$, it is shown that

$$\mathcal{C}(\mathcal{F}) = P_{1,k} + P_{2,k}\mathcal{C}(f_k) + P_{2,k}u(f_k)\mathcal{C}(\mathcal{F}_{k+1}). \tag{21}$$

with initial conditions $\mathcal{C}(\mathcal{F}_{m+1}) = 0$, $P_{1,1} = 0$, $P_{2,1} = 1$ and

$$P_{1,k+1} = P_{1,k} + \mathcal{C}(f_k)P_{2,k} \qquad P_{2,k+1} = P_{2,k}u(f_k). \tag{22}$$

This makes explicit the dependence of the cascade complexity on the complexity of the $k^{th}$ stage.

In practice, $f_k = \sum_l c_l g_l$ for $g_l \in \mathcal{U}$, where $\mathcal{U}$ is a set of functions of approximately identical complexity. For example, the set of projections into Haar features, in which $\mathcal{C}(f_k)$ is proportional to the number of features $g_l$. In general, $f_k$ has three components. The first is a predictor that is also used in a previous cascade stage, e.g. $f_k(x) = f_{k-1}(x) + cg(x)$ for an embedded cascade. In this case, $f_{k-1}(x)$ has already been evaluated in stage $k-1$ and is available with no computational cost. The second is the set $\mathcal{O}(f_k)$ of features that have been used in some stage $j \leq k$. These features are also available and require minimal computation (multiplication by the weight $c_l$ and addition to the running sum). The third is the set $\mathcal{N}(f_k)$ of features that have not been used in any stage $j \leq k$. The overall computation is

$$\mathcal{C}(f_k) = |\mathcal{N}(f_k)| + \lambda|\mathcal{O}(f_k)|, \tag{23}$$

where $\lambda < 1$ is the ratio of computation required to evaluate a used vs. new feature. For Haar wavelets, $\lambda \approx \frac{1}{20}$. It follows that updating the predictor of the $k^{th}$ stage increases its complexity to

$$\mathcal{C}(f_k + \epsilon g) = \begin{cases} \mathcal{C}(f_k) + \lambda & \text{if } g \in \mathcal{O}(f_k) \\ \mathcal{C}(f_k) + 1 & \text{if } g \in \mathcal{N}(f_k), \end{cases} \tag{24}$$

and the complexity of the cascade to

$$\mathcal{C}(\mathcal{F}(f_k + \epsilon g)) \quad = \quad P_{1,k} + P_{2,k}\mathcal{C}(f_k + \epsilon g) + P_{2,k}u(f_k + \epsilon g)\mathcal{C}(\mathcal{F}_{k+1}) \tag{25}$$

$$= \quad \alpha^k + \gamma^k\mathcal{C}(f_k + \epsilon g) + \beta^k u(f_k + \epsilon g) \tag{26}$$

with

$$\alpha^k = P_{1,k} \quad \gamma^k = P_{2,k} \quad \beta^k = P_{2,k}\mathcal{C}(\mathcal{F}_{k+1}). \tag{27}$$

### 3.4 Neutral predictors

The models of (10), (11) and (21) will be used for the design of optimal cascades. Another observation that we will exploit is that

$$\mathcal{H}[\mathcal{F}(f_1, \ldots, f_m, f_m)] = \mathcal{H}[\mathcal{F}(f_1, \ldots, f_m)].$$

This implies that repeating the last stage of a cascade does not change its decision rule. For this reason $n(x) = f_m(x)$ is referred to as the *neutral* predictor of a cascade of $m$ stages.

## 4 Boosting classifier cascades

In this section, we introduce a boosting algorithm for cascade design.

### 4.1 Boosting

Boosting algorithms combine weak learners to produce a complex decision boundary. Boosting iterations are gradient descent steps towards the predictor $f(x)$ of minimum risk for the loss $L[y, f(x)] = e^{-yf(x)}$ [3]. Given a set $\mathcal{U}$ of weak learners, the functional derivative of $R_L$ along the direction of weak leaner $g$ is

$$< \delta R_L(f), g > = \frac{1}{|S_t|} \sum_i \left[ \frac{d}{d\epsilon} e^{-y_i(f(x_i) + \epsilon g(x_i))} \right]_{\epsilon=0} = -\frac{1}{|S_t|} \sum_i y_i w_i g(x_i), \quad (28)$$

where $w_i = e^{-y_i f(x_i)}$ is the weight of $x_i$. Hence, the best update is

$$g^*(x) = \arg\max_{g \in \mathcal{U}} < -\delta R_L(f), g > . \quad (29)$$

Letting $I(x)$ be the indicator function, the optimal step size along the selected direction, $g^*(x)$, is

$$c^* = \arg\min_{c \in R} \sum_i e^{-y_i(f(x_i) + cg^*(x_i))} = \frac{1}{2} \log \frac{\sum_i w_i I(y_i = g^*(x_i))}{\sum_i w_i I(y_i \neq g^*(x_i))}. \quad (30)$$

The predictor is updated into $f(x) = f(x) + c^* g^*(x)$ and the procedure iterated.

### 4.2 Cascade risk minimization

To derive a boosting algorithm for the design of detector cascades, we adopt the loss $L[y, \mathcal{F}(f_1, \ldots, f_m)(x)] = e^{-y\mathcal{F}(f_1, \ldots, f_m)(x)}$, and minimize the cascade risk

$$R_L(\mathcal{F}) = E_{X,Y}\{e^{-y\mathcal{F}(f_1, \ldots, f_m)}\} \approx \frac{1}{|S_t|} \sum_i e^{-y_i \mathcal{F}(f_1, \ldots, f_m)(x_i)}.$$

Using (13) and (19),

$$< \delta R_L(\mathcal{F}(f_k)), g > = \frac{1}{|S_t|} \sum_i \left[ \frac{d}{d\epsilon} e^{-y_i[a^k(x_i) + \epsilon b^k(x_i) g(x_i)]} \right]_{\epsilon=0} = -\frac{1}{|S_t|} \sum_i y_i w_i^k b_i^k g(x_i) \quad (31)$$

where $w_i^k = e^{-y_i a^k(x_i)}$, $b_i^k = b^k(x_i)$ and $a^k, b^k$ are given by (14), (18), and (20). The optimal descent direction and step size for the $k^{th}$ stage are then

$$g_k^* = \arg\max_{g \in \mathcal{U}} < -\delta R_L(\mathcal{F}(f_k)), g > \quad (32)$$

$$c_k^* = \arg\min_{c \in R} \sum_i w_i^k \, e^{-y_i b_i^k c g_k^*(x_i)}. \quad (33)$$

In general, because the $b_i^k$ are not constant, there is no closed form for $c_k^*$, and a line search must be used. Note that, since $a^k(x_i) = \mathcal{F}(f_k)(x_i)$, the weighting mechanism is identical to that of boosting, i.e. points are reweighed according to how well they are classified by the current cascade. Given the optimal $c^*, g^*$ for all stages, the impact of each update in the overall cascade risk, $R_L$, is evaluated and the stage of largest impact is updated.

### 4.3 Adding a new stage

Searching for the optimal cascade configuration requires support for the addition of new stages, whenever necessary. This is accomplished by including a neutral predictor as the last stage of the cascade. If adding a weak learner to the neutral stage reduces the risk further than the corresponding addition to any other stage, a new stage (containing the neutral predictor plus the weak learner) is created. Since this new stage includes the last stage of the previous cascade, the process mimics the design of an embedded cascade. However, there are no restrictions that a new stage should be added at each boosting iteration, or consist of a single weak learner.

### 4.4 Incorporating complexity constraints

Joint optimization of speed and accuracy, requires the minimization of the Lagrangian of (4). This requires the computation of the functional derivatives

$$< \delta R_C(\mathcal{F}(f_k)), g > = \frac{1}{|S_t^-|} \sum_i y_i^s \left\{ \frac{d}{d\epsilon} L_C[\mathcal{C}(\mathcal{F}(f_k + \epsilon g)(x_i)] \right\}_{\epsilon=0} \tag{34}$$

where $y_i^s = I(y_i = -1)$. Similarly to boosting, which upper bounds the zero-one loss $u(-yf)$ by the exponential loss $e^{-yf}$, we rely on a loss that upper-bounds the true complexity. This upper-bound is a combination of a boosting-style bound $u(f + \epsilon g) \le e^{f+\epsilon g}$, and the bound $\mathcal{C}(f + \epsilon g) \le \mathcal{C}(f) + 1$, which follows from (24). Using (26),

$$L_C[\mathcal{C}(\mathcal{F}(f_k + \epsilon g)(x_i)] = L_C[\alpha^k + \gamma^k \mathcal{C}(f_k + \epsilon g) + \beta^k u(f_k + \epsilon g)] \tag{35}$$
$$= \alpha^k + \gamma^k(\mathcal{C}(f_k) + 1) + \beta^k e^{f_k + \epsilon g} \tag{36}$$

and, since $\left\{ \frac{d}{d\epsilon} L_C[\mathcal{C}(\mathcal{F}(f_k + \epsilon g))] \right\}_{\epsilon=0} = \beta^k e^{f_k} g$,

$$< \delta R_C(\mathcal{F}(f_k)), g > = \frac{1}{|S_t^-|} \sum_i y_i^s \psi_i^k \beta_i^k g(x_i) \tag{37}$$

with $\beta_i^k = \beta^k(x_i)$ and $\psi_i^k = e^{f_k(x_i)}$. The derivative of (4) with respect to the $k^{th}$ stage predictor is then

$$< \delta \mathcal{L}(\mathcal{F}(f_k)), g > = < \delta R_L(\mathcal{F}(f_k)), g > + \eta < \delta R_C(\mathcal{F}(f_k)), g > \tag{38}$$
$$= \sum_i \left( -\frac{y_i w_i^k b_i^k}{|S_t|} + \eta \frac{y_i^s \psi_i^k \beta_i^k}{|S_t^-|} \right) g(x_i) \tag{39}$$

with $w_i^k = e^{-y_i a^k(x_i)}$ and $a^k$ and $b^k$ given by (14), (18), and (20). Given a set of weak learners $\mathcal{U}$, the optimal descent direction and step size for the $k^{th}$ stage are then

$$g_k^* = \arg\max_{g \in \mathcal{U}} < -\delta \mathcal{L}(\mathcal{F}(f_k)), g > \tag{40}$$

$$c_k^* = \arg\min_{c \in R} \left\{ \frac{1}{|S_t|} \sum_i w_i^k e^{-y_i b_i^k c g_k^*(x_i)} + \frac{\eta}{|S_t^-|} \sum_i y_i^s \psi_i^k \beta_i^k e^{c g_k^*(x_i)} \right\}. \tag{41}$$

A pair $(g_{k,1}^*, c_{k,1}^*)$ is found among the set $\mathcal{O}(f_k)$ and another among the set $\mathcal{U} - \mathcal{O}(f_k)$. The one that most reduces (4) is selected as the best update for the $k^{th}$ stage and the stage with the largest impact is updated. This gradient descent procedure is denoted *Fast Cascade Boosting* (FCBoost).

## 5 Extensions

FCBoost supports a number of extensions that we briefly discuss in this section.

### 5.1 Cost Sensitive Boosting

As is the case for AdaBoost, it is possible to use cost sensitive risks in FCBoost. For example, the risk of *CS-AdaBoost*: $R_L(f) = E_{X,Y}\{y^c e^{-yf(x)}\}$ [12] or *Asym-AdaBoost*: $R_L(f) = E_{X,Y}\{e^{-y^c yf(x)}\}$ [8], where $y^c = CI(y = -1) + (1 - C)I(y = 1)$ and $C$ is a cost factor.

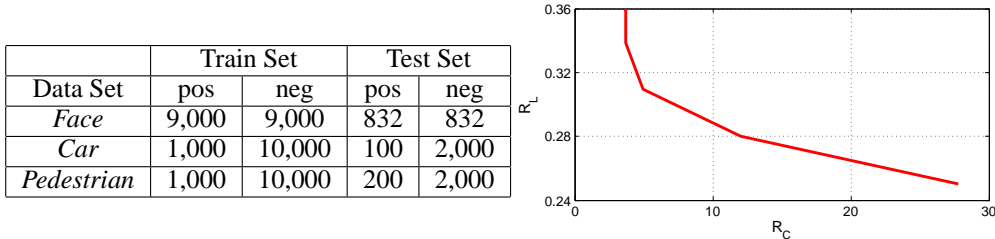

| Data Set | Train Set | | Test Set | |
|---|---|---|---|---|
| | pos | neg | pos | neg |
| *Face* | 9,000 | 9,000 | 832 | 832 |
| *Car* | 1,000 | 10,000 | 100 | 2,000 |
| *Pedestrian* | 1,000 | 10,000 | 200 | 2,000 |

Figure 2: Left: data set characteristics. Right: Trade-off between the error ($R_L$) and complexity ($R_C$) components of the risk as $\eta$ changes in (4).

Table 1: Performance of various classifiers on the face, car, and pedestrian test sets.

| Method | Face | | | Car | | | Pedestrian | | |
|---|---|---|---|---|---|---|---|---|---|
| | $R_L$ | $R_C$ | $\mathcal{L}$ | $R_L$ | $R_C$ | $\mathcal{L}$ | $R_L$ | $R_C$ | $\mathcal{L}$ |
| *AdaBoost* | **0.20** | 50 | 1.20 | **0.22** | 50 | 1.22 | **0.35** | 50 | 1.35 |
| *ChainBoost* | 0.45 | **2.65** | 0.50 | 0.65 | **2.40** | 0.70 | .052 | **3.34** | 0.59 |
| *FCBoost ($\eta = 0.02$)* | 0.30 | 4.93 | **0.40** | 0.44 | 5.38 | **0.55** | 0.46 | 4.23 | **0.54** |

## 5.2 Bootstrapping

Bootstrapping is a procedure to augment the training set, by using false positives of the current classifier as the training set for the following [11]. This improves performance, but is feasible only when the bootstrapping procedure *does not* affect previously rejected examples. Otherwise, the classifier will forget the previous negatives while learning from the new ones. Since FCBoost learns all cascade stages simultaneously, and any stage can change after bootstrapping, this condition is violated. To overcome the problem, rather than replacing all negative examples with false positives, only a random subset is replaced. The negatives that remain in the training set prevent the classifier from forgetting about the previous iterations. This method is used to update the training set whenever the false positive rate of the cascade being learned reaches $50\%$.

## 6 Evaluation

Several experiments were performed to evaluate the performance of FCBoost, using face, car, and pedestrian recognition data sets, from computer vision. In all cases, Haar wavelet features were used as weak learners. Figure 2 summarizes the data sets.

**Effect of $\eta$:** We started by measuring the impact of $\eta$, see (4), on the accuracy and complexity of FCBoost cascades. Figure 2 plots the accuracy component of the risk, $R_L$, as a function of the complexity component, $R_C$, on the face data set, for cascades trained with different $\eta$. The leftmost point corresponds to $\eta = 0.05$, and the rightmost to $\eta = 0$. As expected, as $\eta$ decreases the cascade has lower error and higher complexity. In the remaining experiments we used $\eta = 0.02$.

**Cascade comparison:** Figure 3 (a) repeats the plots of the Lagrangian of the risk shown in Figure 1, for classifiers trained with 50 boosting iterations, on the face data. In addition to AdaBoost and ChainBoost, it presents the curves of FCBoost with ($\eta = 0.02$) and without ($\eta = 0$) complexity constraints. Note that, in the latter case, performance is in between those of AdaBoost and ChainBoost. This reflects the fact that FCBoost ($\eta = 0$) does produce a cascade, but this cascade has worse accuracy/complexity trade-off than that of ChainBoost. On the other hand, the inclusion of complexity constraints, FCBoost ($\eta = 0.02$), produces a cascade with the best trade-off. These results are confirmed by Table 1, which compares classifiers trained on all data sets. In all cases, AdaBoost detectors have the lowest error, but at a tremendous computational cost. On the other hand, ChainBoost cascades are always the fastest, at the cost of the highest classification error. Finally, FCBoost ($\eta = 0.02$) achieves the best accuracy/complexity trade-off: its cascade has the lowest risk Lagrangian $\mathcal{L}$. It is close to ten times faster than the AdaBoost detector, and has half of the increase in classification error (with respect to AdaBoost) of the ChainBoost cascade. Based on these results, FCBoost ($\eta = 0.02$) was used in the last experiment.

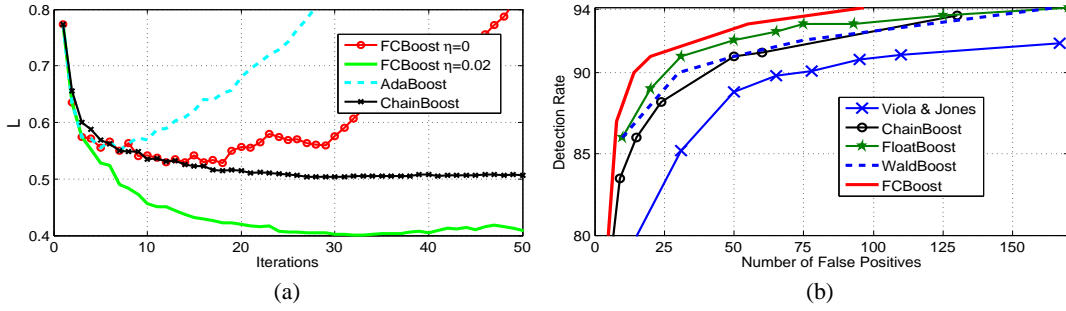

Figure 3: a) Lagrangian of the risk for classifiers trained with various boosting algorithms. b) ROC of various detector cascades on the MIT-CMU data set.

Table 2: Comparison of the speed of different detectors.

| Method | VJ [13] | FloatBoost [5] | ChainBoost [15] | WaldBoost [9] | [8] | FCBoost |
|--------|---------|----------------|-----------------|---------------|------|---------|
| Evals | 8 | 18.9 | 18.1 | 10.84 | 15.45 | **7.2** |

**Face detection:** We finish with a face detector designed with FCBoost ($\eta = 0.02$), bootstrapping, and $130K$ Haar features. To make the detector cost-sensitive, we used *CS-AdaBoost* with $C = 0.99$. Figure 3 b) compares the resulting ROC to those of VJ [13], ChainBoost [15], FloatBoost [5] and WaldBoost [9]. Table 2 presents a similar comparison for the detector speed (average number of features evaluated per patch). Note the superior performance of the FCBoost cascade in terms of *both* accuracy and speed. To the best of our knowledge, this is the fastest face detector reported to date.

## A Recursive form of cascade predictor

Applying (9) recursively to (8)

$$
\begin{align}
\mathcal{F} &= f_1 u(-f_1) + u(f_1)\mathcal{F}_2 \tag{42} \\
&= f_1 u(-f_1) + u(f_1)\left[f_2 u(-f_2) + u(f_2)\mathcal{F}_3\right] \tag{43} \\
&= f_1 u(-f_1) + f_2 u(f_1)u(-f_2) + u(f_1)u(f_2)\left[f_3 u(-f_3) + u(f_3)\mathcal{F}_4\right] \tag{44} \\
&= \sum_{i=1}^{k-1} f_i u(-f_i) \prod_{j<i} u(f_j) + \mathcal{F}_k \prod_{j<k} u(f_j) \tag{45} \\
&= T_{1,k} + T_{2,k}\mathcal{F}_k \tag{46}
\end{align}
$$

where $T_{1,k} = \sum_{i=1}^{k-1} f_i u(-f_i) \prod_{j<i} u(f_j)$ and $T_{2,k} = \prod_{j<k} u(f_j)$ satisfy the recursions of (12). Combining (46) and (9) then leads to (11). (10) follows from (46) and the initial condition $\mathcal{F}_m = f_m$.

## B Recursive form of cascade complexity

Let $\mathcal{C}(f_k)$ be the complexity of evaluating $f_k$. Then

$$
\begin{align}
\mathcal{C}(\mathcal{F}) &= \mathcal{C}(f_1) + u(f_1)\mathcal{C}(\mathcal{F}_2) \tag{47} \\
&= \mathcal{C}(f_1) + u(f_1)[\mathcal{C}(f_2) + u(f_2)\mathcal{C}(\mathcal{F}_3)] \tag{48} \\
&= \sum_{i=1}^{k-1} \mathcal{C}(f_i) \prod_{j<i} u(f_j) + \mathcal{C}(\mathcal{F}_k) \prod_{j<k} u(f_j) \tag{49} \\
&= P_{1,k} + P_{2,k}\mathcal{C}(\mathcal{F}_k) \tag{50}
\end{align}
$$

with

$$
P_{1,k+1} = P_{1,k} + \mathcal{C}(f_k)\,P_{2,k} \qquad P_{2,k+1} = P_{2,k}\,u(f_k) \tag{51}
$$

and initial conditions $P_{1,1} = 0$, $P_{2,1} = 1$. The relationship of (47) is a special case of

$$
\mathcal{C}(\mathcal{F}_k) = \mathcal{C}(f_k) + u(f_k)\mathcal{C}(\mathcal{F}_{k+1}) \tag{52}
$$

with initial conditions $\mathcal{C}(\mathcal{F}_m) = \mathcal{C}(f_m)$ and $\mathcal{C}(\mathcal{F}_{m+1}) = 0$. Combining (52) with (50) leads to (21).

# References

[1] S. C. Brubaker, M. D. Mullin, and J. M. Rehg. On the design of cascades of boosted ensembles for face detection. *International Journal of Computer Vision*, 77:65–86, 2008.

[2] Y. Freund and R. E. Schapire. A decision-theoretic generalization of on-line learning and an application to boosting, 1997.

[3] J. Friedman, T. Hastie, and R. Tibshirani. Additive logistic regression: a statistical view of boosting. *Annals of Statistics*, 28:2000, 1998.

[4] X. Hou, C.-L. Liu, and T. Tan. Learning boosted asymmetric classifiers for object detection. In *IEEE Conference on Computer Vision and Pattern Recognition,*, pages 330–338, 2006.

[5] S. Z. Li and Z. Zhang. Floatboost learning and statistical face detection. *IEEE Trans. on Pattern Analysis and Machine Intelligence*, 26(9):1112–1123, 2004.

[6] C. Liu and H.-Y. Shum;. Kullback-leibler boosting. In *IEEE Conference on Computer Vision and Pattern Recognition*, pages 587–594, 2003.

[7] H. Luo. Optimization design of cascaded classifiers. In *IEEE Conference on Computer Vision and Pattern Recognition,*, pages 480–485, 2005.

[8] H. Masnadi-Shirazi and N. Vasconcelos. High detection-rate cascades for real-time object detection. In *IEEE International Conference on Computer Vision*, volume 2, pages 1–6, 2007.

[9] J. Sochman and J. Matas. Waldboost - learning for time constrained sequential detection. In *IEEE Conference on Computer Vision and Pattern Recognition*, pages 150–157, 2005.

[10] J. Sun, J. M. Rehg, and A. Bobick. Automatic cascade training with perturbation bias. *IEEE Conference on Computer Vision and Pattern Recognition*, 2:276–283, 2004.

[11] K. K. Sung and T. Poggio. Example based learning for view-based human face detection. *IEEE Trans. on Pattern Analysis and Machine Intelligence*, 20:39–51, 1998.

[12] P. Viola and M. Jones. Fast and robust classification using asymmetric adaboost and a detector cascade. In *Advances in Neural Information Processing System*, pages 1311–1318, 2001.

[13] P. Viola and M. Jones. Robust real-time object detection. *International Journal of Computer Vision*, 57(2):137–154, 2004.

[14] J. Wu, S. Brubaker, M. D. Mullin, and J. M. Rehg. Fast asymmetric learning for cascade face detection. *IEEE Trans. on Pattern Analysis and Machine Intelligence*, 3:369–382, 2008.

[15] R. Xiao, L. Zhu, and H.-J. Zhang. Boosting chain learning for object detection. In *IEEE International Conference on Computer Vision*, pages 709–715, 2003.

